# Active Estimation of F-Measures

**Christoph Sawade, Niels Landwehr, and Tobias Scheffer**
University of Potsdam
Department of Computer Science
August-Bebel-Strasse 89, 14482 Potsdam, Germany
{sawade, landwehr, scheffer}@cs.uni-potsdam.de

## Abstract

We address the problem of estimating the $F_\alpha$-measure of a given model as accurately as possible on a fixed labeling budget. This problem occurs whenever an estimate cannot be obtained from held-out training data; for instance, when data that have been used to train the model are held back for reasons of privacy or do not reflect the test distribution. In this case, new test instances have to be drawn and labeled at a cost. An active estimation procedure selects instances according to an instrumental sampling distribution. An analysis of the sources of estimation error leads to an optimal sampling distribution that minimizes estimator variance. We explore conditions under which active estimates of $F_\alpha$-measures are more accurate than estimates based on instances sampled from the test distribution.

## 1 Introduction

This paper addresses the problem of evaluating a given model in terms of its predictive performance. In practice, it is not always possible to evaluate a model on held-out training data; consider, for instance, the following scenarios. When a readily trained model is shipped and deployed, training data may be held back for reasons of privacy. Secondly, training data may have been created under laboratory conditions and may not entirely reflect the test distribution. Finally, when a model has been trained actively, the labeled data is biased towards small-margin instances which would incur a pessimistic bias on any cross-validation estimate.

This problem has recently been studied for risks—*i.e.,* for performance measures which are integrals of a loss function over an instance space [7]. However, several performance measures cannot be expressed as a risk. Perhaps the most prominent such measure is the $F_\alpha$-measure [10]. For a given binary classifier and sample of size $n$, let $n_{tp}$ and $n_{fp}$ denote the number of true and false positives, respectively, and $n_{fn}$ the number of false negatives. Then the classifier's $F_\alpha$-measure on the sample is defined as

$$F_\alpha = \frac{n_{tp}}{\alpha(n_{tp} + n_{fp}) + (1 - \alpha)(n_{tp} + n_{fn})}. \tag{1}$$

Precision and recall are special cases for $\alpha = 1$ and $\alpha = 0$, respectively. The $F_\alpha$-measure is defined as an estimator in terms of empirical quantities. This is unintuitive from a statistical point of view and raises the question which quantity of the underlying distribution the $F$-measure actually estimates. We will now introduce the class of generalized risk functionals that we study in this paper. We will then show that $F_\alpha$ is a consistent estimate of a quantity that falls into this class.

Let $\mathcal{X}$ denote the feature space and $\mathcal{Y}$ the label space. An unknown test distribution $p(\mathbf{x}, y)$ is defined over $\mathcal{X} \times \mathcal{Y}$. Let $p(y|\mathbf{x}; \theta)$ be a given $\theta$-parameterized model of $p(y|\mathbf{x})$ and let $f_\theta : \mathcal{X} \to \mathcal{Y}$ with $f_\theta(\mathbf{x}) = \arg\max_y p(y|\mathbf{x}; \theta)$ be the corresponding hypothesis.

Like any risk functional, the generalized risk is parameterized with a function $\ell : \mathcal{Y} \times \mathcal{Y} \to \mathbb{R}$ determining either the loss or—alternatively—the gain that is incurred for a pair of predicted and

true label. In addition, the generalized risk is parameterized with a function $w$ that assigns a weight $w(\mathbf{x}, y, f_\theta) = 1$ to each instance. For instance, precision sums over instances with $f_\theta(\mathbf{x}) = 1$ with weight 1 and gives no consideration to other instances. Equation 2 defines the generalized risk:

$$G = \frac{\iint \ell(f_\theta(\mathbf{x}), y) w(\mathbf{x}, y, f_\theta) p(\mathbf{x}, y) dy d\mathbf{x}}{\iint w(\mathbf{x}, y, f_\theta) p(\mathbf{x}, y) dy d\mathbf{x}}. \tag{2}$$

The integral over $\mathcal{Y}$ is replaced by a sum in the case of a discrete label space $\mathcal{Y}$. Note that the generalized risk (Equation 2) reduces to the regular risk for $w(\mathbf{x}, y, f_\theta) = 1$. On a sample of size $n$, a consistent estimator can be obtained by replacing the cumulative distribution function with the empirical distribution function.

**Proposition 1.** *Let* $(\mathbf{x}_1, y_1), \ldots, (\mathbf{x}_n, y_n)$ *be drawn* iid *according to* $p(\mathbf{x}, y)$. *The quantity*

$$\hat{G}_n = \frac{\sum_{i=1}^{n} \ell(f_\theta(\mathbf{x}_i), y_i) w(\mathbf{x}_i, y_i, f_\theta)}{\sum_{i=1}^{n} w(\mathbf{x}_i, y_i, f_\theta)} \tag{3}$$

*is a consistent estimate of the generalized risk* $G$ *defined by Equation 2.*

*Proof.* The proposition follows from Slutsky's theorem [3] applied to the numerator and denominator of Equation 3. □

Consistency means asymptotical unbiasedness; that is, the expected value of the estimate $\hat{G}_n$ converges in distribution to the true risk $G$ for $n \to \infty$. We now observe that $F_\alpha$-measures—including precision and recall—are consistent empirical estimates of generalized risks for appropriately chosen functions $w$.

**Corollary 1.** $F_\alpha$ *is a consistent estimate of the generalized risk with* $\mathcal{Y} = \{0, 1\}$, $w(\mathbf{x}, y, f_\theta) = \alpha f_\theta(\mathbf{x}) + (1-\alpha)y$ *and* $\ell = 1 - \ell_{0/1}$, *where* $\ell_{0/1}$ *denotes the zero-one loss.*

*Proof.* The claim follows from Proposition 1 since

$$\hat{G}_n = \frac{\sum_{i=1}^{n}(1 - \ell_{0/1}(f_\theta(\mathbf{x}_i), y_i))\left(\alpha f_\theta(\mathbf{x}_i) + (1-\alpha)y_i\right)}{\sum_{i=1}^{n}\left(\alpha f_\theta(\mathbf{x}_i) + (1-\alpha)y_i\right)}$$

$$= \frac{\sum_{i=1}^{n} f_\theta(\mathbf{x}_i)y_i}{\alpha \sum_{i=1}^{n} f_\theta(\mathbf{x}_i) + (1-\alpha)\sum_{i=1}^{n} y_i} = \frac{n_{tp}}{\alpha\left(n_{tp} + n_{fp}\right) + (1-\alpha)\left(n_{tp} + n_{fn}\right)}. \quad □$$

Having established and motivated the generalized risk functional, we now turn towards the problem of acquiring a consistent estimate with minimal estimation error on a fixed labeling budget $n$. Test instances $\mathbf{x}_1, ..., \mathbf{x}_n$ need not necessarily be drawn according to the distribution $p$. Instead, we study an *active estimation process* that selects test instances according to an instrumental distribution $q$. When instances are sampled from $q$, an estimator of the generalized risk can be defined as

$$\hat{G}_{n,q} = \frac{\sum_{i=1}^{n} \frac{p(\mathbf{x}_i)}{q(\mathbf{x}_i)} \ell(f_\theta(\mathbf{x}_i), y_i) w(\mathbf{x}_i, y_i, f_\theta)}{\sum_{i=1}^{n} \frac{p(\mathbf{x}_i)}{q(\mathbf{x}_i)} w(\mathbf{x}_i, y_i, f_\theta)} \tag{4}$$

where $(\mathbf{x}_i, y_i)$ are drawn from $q(\mathbf{x})p(y|\mathbf{x})$. Weighting factors $\frac{p(\mathbf{x}_i)}{q(\mathbf{x}_i)}$ compensate for the discrepancy between test and instrumental distributions. Because of the weighting factors, Slutsky's Theorem again implies that Equation 4 defines a consistent estimator for $G$, under the precondition that for all $\mathbf{x} \in \mathcal{X}$ with $p(\mathbf{x}) > 0$ it holds that $q(\mathbf{x}) > 0$. Note that Equation 3 is a special case of Equation 4, using the instrumental distribution $q = p$.

The estimate $\hat{G}_{n,q}$ given by Equation 4 depends on the selected instances $(\mathbf{x}_i, y_i)$, which are drawn according to the distribution $q(\mathbf{x})p(y|\mathbf{x})$. Thus, $\hat{G}_{n,q}$ is a random variable whose distribution depends on $q$. Our overall goal is to determine the instrumental distribution $q$ such that the expected deviation from the generalized risk is minimal for fixed labeling costs $n$:

$$q^* = \arg\min_q \mathbb{E}\left[\left(\hat{G}_{n,q} - G\right)^2\right].$$

## 2 Active Estimation through Variance Minimization

The bias-variance decomposition expresses the estimation error as a sum of a squared bias and a variance term [5]:

$$\mathbb{E}\left[(\hat{G}_{n,q} - G)^2\right] = \left(\mathbb{E}\left[\hat{G}_{n,q}\right] - G\right)^2 + \mathbb{E}\left[\left(\hat{G}_{n,q} - \mathbb{E}\left[\hat{G}_{n,q}\right]\right)^2\right] \tag{5}$$

$$= \text{Bias}^2[\hat{G}_{n,q}] + \text{Var}[\hat{G}_{n,q}]. \tag{6}$$

Because $\hat{G}_{n,q}$ is consistent, both $\text{Bias}^2[\hat{G}_{n,q}]$ and $\text{Var}[\hat{G}_{n,q}]$ vanish for $n \to \infty$. More specifically, Lemma 1 shows that $\text{Bias}^2[\hat{G}_{n,q}]$ is of order $\frac{1}{n^2}$.

**Lemma 1** (Bias of Estimator). *Let $\hat{G}_{n,q}$ be as defined in Equation 4. Then there exists $C \geq 0$ with*

$$\left|\mathbb{E}\left[\hat{G}_{n,q}\right] - G\right| \leq \frac{C}{n}. \tag{7}$$

The proof can be found in the online appendix. Lemma 2 states that the active risk estimator $\hat{G}_{n,q}$ is asymptotically normally distributed, and characterizes its variance in the limit.

**Lemma 2** (Asymptotic Distribution of Estimator). *Let $\hat{G}_{n,q}$ be defined as in Equation 4. Then,*

$$\sqrt{n}\left(\hat{G}_{n,q} - G\right) \overset{n\to\infty}{\longrightarrow} \mathcal{N}\left(0, \sigma_q^2\right) \tag{8}$$

*with asymptotic variance*

$$\sigma_q^2 = \int \frac{p(\mathbf{x})}{q(\mathbf{x})} \left(\int w(\mathbf{x}, y, f_\theta)^2 \left(\ell(f_\theta(\mathbf{x}), y) - G\right)^2 p(y|\mathbf{x})dy\right) p(\mathbf{x})d\mathbf{x} \tag{9}$$

*where $\overset{n\to\infty}{\longrightarrow}$ denotes convergence in distribution.*

A proof of Lemma 2 can be found in the appendix. Taking the variance of Equation 8, we obtain

$$n \text{Var}\left[\hat{G}_{n,q}\right] \overset{n\to\infty}{\longrightarrow} \sigma_q^2, \tag{10}$$

thus $\text{Var}[\hat{G}_{n,q}]$ is of order $\frac{1}{n}$. As the bias term vanishes with $\frac{1}{n^2}$, the expected estimation error $\mathbb{E}[(\hat{G}_{n,q} - G)^2]$ will be dominated by $\text{Var}[\hat{G}_{n,q}]$. Moreover, Equation 10 indicates that $\text{Var}[\hat{G}_{n,q}]$ can be approximately minimized by minimizing $\sigma_q^2$. In the following, we will consequently derive a sampling distribution $q^*$ that minimizes the asymptotic variance $\sigma_q^2$ of the estimator $\hat{G}_{n,q}$.

### 2.1 Optimal Sampling Distribution

The following theorem derives the sampling distribution that minimizes the asymptotic variance $\sigma_q^2$:

**Theorem 1** (Optimal Sampling Distribution). *The instrumental distribution that minimizes the asymptotic variance $\sigma_q^2$ of the generalized risk estimator $\hat{G}_{n,q}$ is given by*

$$q^*(\mathbf{x}) \propto p(\mathbf{x})\sqrt{\int w(\mathbf{x}, y, f_\theta)^2 \left(\ell(f_\theta(\mathbf{x}), y) - G\right)^2 p(y|\mathbf{x})dy}. \tag{11}$$

A proof of Theorem 1 is given in the appendix. Since $F$-measures are estimators of generalized risks according to Corollary 1, we can now derive their variance-minimizing sampling distributions.

**Corollary 2** (Optimal Sampling for $F_\alpha$). *The sampling distribution that minimizes the asymptotic variance of the $F_\alpha$-estimator resolves to*

$$q^*(\mathbf{x}) \propto \begin{cases} p(\mathbf{x})\sqrt{p(f_\theta(\mathbf{x})|\mathbf{x})(1-G)^2 + \alpha^2(1 - p(f_\theta(\mathbf{x})|\mathbf{x}))G^2} & : f(\mathbf{x}) = 1 \\ p(\mathbf{x})(1-\alpha)\sqrt{(1 - p(f_\theta(\mathbf{x})|\mathbf{x}))G^2} & : f(\mathbf{x}) = 0 \end{cases} \tag{12}$$

---

**Algorithm 1** Active Estimation of $F_\alpha$-Measures

---

**input** Model parameters $\theta$, pool $D$, labeling costs $n$.
**output** Generalized risk estimate $\hat{G}_{n,q^*}$.

1: Compute optimal sampling distribution $q^*$ according to Corollary 2, 3, or 4, respectively.
2: **for** $i = 1, \ldots, n$ **do**
3:   Draw $\mathbf{x}_i \sim q^*(\mathbf{x})$ from $D$ with replacement.
4:   Query label $y_i \sim p(y|\mathbf{x}_i)$ from oracle.
5: **end for**
6: **return** $\dfrac{\sum_{i=1}^n \frac{1}{q(\mathbf{x}_i)} \ell(f_\theta(\mathbf{x}_i), y_i) w(\mathbf{x}_i, y_i, f_\theta)}{\sum_{i=1}^n \frac{1}{q(\mathbf{x}_i)} w(\mathbf{x}_i, y_i, f_\theta)}$

---

*Proof.* According to Corollary 1, $F_\alpha$ estimates a generalized risk with $\mathcal{Y} = \{0, 1\}$, $w(\mathbf{x}, y, f_\theta) = \alpha f_\theta(\mathbf{x}) + (1-\alpha)y$ and $\ell = 1 - \ell_{0/1}$. Starting from Theorem 1, we derive

$$q^*(\mathbf{x}) \propto p(\mathbf{x}) \sqrt{\sum_{y \in \{0,1\}} \left( \alpha f_\theta(\mathbf{x}) + (1-\alpha)y \right)^2 \left( 1 - \ell_{0/1}(f_\theta(\mathbf{x}), y) - G \right)^2 p(y|\mathbf{x})} \tag{13}$$

$$= p(\mathbf{x}) \left( \alpha^2 f_\theta(\mathbf{x}) \left( (1 - f_\theta(\mathbf{x})) - G \right)^2 p(y=0|\mathbf{x}) \right.$$

$$\left. + (1 - \alpha(1 - f_\theta(\mathbf{x})))^2 \left( f_\theta(\mathbf{x}) - G \right)^2 p(y=1|\mathbf{x}) \right)^{\frac{1}{2}} \tag{14}$$

The claim follows by case differentiation according to the value of $f_\theta(\mathbf{x})$. $\square$

**Corollary 3** (Optimal Sampling for Recall). *The sampling distribution that minimizes $\sigma_q^2$ for recall resolves to*

$$q^*(\mathbf{x}) \propto \begin{cases} p(\mathbf{x}) \sqrt{p(f_\theta(\mathbf{x})|\mathbf{x})(1-G)^2} & : f(\mathbf{x}) = 1 \\ p(\mathbf{x}) \sqrt{(1 - p(f_\theta(\mathbf{x})|\mathbf{x}))G^2} & : f(\mathbf{x}) = 0. \end{cases} \tag{15}$$

**Corollary 4** (Optimal Sampling for Precision). *The sampling distribution that minimizes $\sigma_q^2$ for precision resolves to*

$$q^*(\mathbf{x}) \propto p(\mathbf{x}) f_\theta(\mathbf{x}) \sqrt{(1 - 2G)p(f_\theta(\mathbf{x})|\mathbf{x}) + G^2}. \tag{16}$$

Corollaries 3 and 4 directly follow from Corollary 2 for $\alpha = 0$ and $\alpha = 1$. Note that for standard risks (that is, $w = 1$) Theorem 1 coincides with the optimal sampling distribution derived in [7].

### 2.2 Empirical Sampling Distribution

Theorem 1 and Corollaries 2, 3, and 4 depend on the unknown test distribution $p(\mathbf{x})$. We now turn towards a setting in which a large pool $D$ of unlabeled test instances is available. Instances from this pool can be sampled and then labeled at a cost. Drawing instances from the pool replaces generating them under the test distribution; that is, $p(\mathbf{x}) = \frac{1}{m}$ for all $\mathbf{x} \in D$.

Theorem 1 and its corollaries also depend on the true conditional $p(y|\mathbf{x})$. To implement the method, we have to approximate the true conditional $p(y|\mathbf{x})$; we use the model $p(y|\mathbf{x}; \theta)$. This approximation constitutes an analogy to active learning: In active learning, the model-based output probability $p(y|\mathbf{x}; \theta)$ serves as the basis on which the least confident instances are selected. Note that as long as $p(\mathbf{x}) > 0$ implies $q(\mathbf{x}) > 0$, the weighting factors ensure that such approximations do not introduce an asymptotic bias in our estimator (Equation 4). Finally, Theorem 1 and its corollaries depend on the true generalized risk $G$. $G$ is replaced by an intrinsic generalized risk calculated from Equation 2, where the integral over $\mathcal{X}$ is replaced by a sum over the pool, $p(\mathbf{x}) = \frac{1}{m}$, and $p(y|\mathbf{x}) \approx p(y|\mathbf{x}; \theta)$.

Algorithm 1 summarizes the procedure for active estimation of $F$-measures. A special case occurs when the labeling process is deterministic. Since instances are sampled with replacement, elements may be drawn more than once. In this case, labels can be looked up rather than be queried from the deterministic labeling oracle repeatedly. The loop may then be continued until the labeling budget is exhausted. Note that $F$-measures are undefined when the denominator is zero which is the case when all drawn examples have a weight $w$ of zero. For instance, precision is undefined when no positive examples have been drawn.

## 2.3 Confidence Intervals

Lemma 2 shows that the estimator $\hat{G}_{n,q}$ is asymptotically normally distributed and characterizes its asymptotic variance. A consistent estimate of $\sigma_q^2$ is obtained from the labeled sample $(\mathbf{x}_1, y_1), \ldots, (\mathbf{x}_n, y_n)$ drawn from the distribution $q(\mathbf{x})p(y|\mathbf{x})$ by computing empirical variance

$$S_{n,q}^2 = \frac{1}{\sum_{i=1}^n \frac{p(\mathbf{x}_i)}{q(\mathbf{x}_i)}} \sum_{i=1}^n \left( \frac{p(\mathbf{x}_i)}{q(\mathbf{x}_i)} \right)^2 w(\mathbf{x}_i, y_i, f_\theta)^2 \left( \ell(f_\theta(\mathbf{x}_i), y_i) - \hat{G}_{n,q} \right)^2.$$

A two-sided confidence interval $[\hat{G}_{n,q} - z, \hat{G}_{n,q} + z]$ with coverage $1 - \rho$ is now given by $z = F_n^{-1}\left(1 - \frac{\rho}{2}\right) \frac{S_{n,q}}{\sqrt{n}}$ where $F_n^{-1}$ is the inverse cumulative distribution function of the Student's $t$ distribution. As in the standard case of drawing test instances $\mathbf{x}_i$ from the original distribution $p$, such confidence intervals are approximate for finite $n$, but become exact for $n \to \infty$.

# 3 Empirical Studies

We compare active estimation of $F_\alpha$-measures according to Algorithm 1 (denoted $active_F$) to estimation based on a sample of instances drawn uniformly from the pool (denoted $passive$). We also consider the active estimator for risks presented in [7]. Instances are drawn according to the optimal sampling distribution $q_{0/1}^*$ for zero-one risk (Derivation 1 in [7]); the $F_\alpha$-measure is computed according to Equation 4 using $q = q_{0/1}^*$ (denoted $active_{err}$).

## 3.1 Experimental Setting and Domains

For each experimental domain, data is split into a training set and a pool of test instances. We train a kernelized regularized logistic regression model $p(y|\mathbf{x}; \theta)$ (using the implementation of Yamada [11]). All methods operate on identical labeling budgets $n$. The evaluation process is averaged over 1,000 repetitions. In case one of the repetitions results in an undefined estimate, the entire experiment is discarded (*i.e.,* there is no data point for the method in the corresponding diagram).

**Spam filtering domain.** Spammers impose a shift on the distribution over time as they implement new templates and generators. In our experiments, a filter trained in the past has to be evaluated with respect to a present distribution of emails. We collect 169,612 emails from an email service provider between June 2007 and April 2010; of these, 42,165 emails received by February 2008 are used for training. Emails are represented by 541,713 binary bag-of-word features. Approximately 5% of all emails fall into the positive class *non-spam*.

**Text classification domain.** The Reuters-21578 text classification task [4] allows us to study the effect of class skew, and serves as a prototypical domain for active learning. We experiment on the ten most frequently occurring topics. We employ an active learner that always queries the example with minimal functional margin $p(f_\theta(\mathbf{x})|\mathbf{x}; \theta) - \max_{y \neq f_\theta(\mathbf{x})} p(y|\mathbf{x}; \theta)$ [9]. The learning process is initialized with one labeled training instance from each class, another 200 class labels are queried.

**Digit recognition domain.** We also study a digit recognition domain in which training and test data originate from different sources. A detailed description is included in the online appendix.

## 3.2 Empirical Results

We study the performance of active and passive estimates as a function of (a) the precision-recall trade-off parameter $\alpha$, (b) the discrepancy between training and test distribution, and (c) class skew in the test distribution. Point (b) is of interest because active estimates require the approximation $p(y|\mathbf{x}) \approx p(y|\mathbf{x}; \theta)$; this assumption is violated when training and test distributions differ.

**Effect of the trade-off parameter $\alpha$.** For the spam filtering domain, Figure 1 shows the average absolute estimation error for $F_0$ (recall), $F_{0.5}$, and $F_1$ (precision) estimates on a test set of 33,296 emails received between February 2008 and October 2008. The active generalized risk estimate $active_F$ significantly outperforms the passive estimate $passive$ for all three measures. In order to reach the estimation accuracy of $passive$ with a labeling budget of $n = 800$, $active_F$ requires fewer than 150 (recall), 200 ($F_{0.5}$), or 100 (precision) labeled test instances. Estimates obtained from

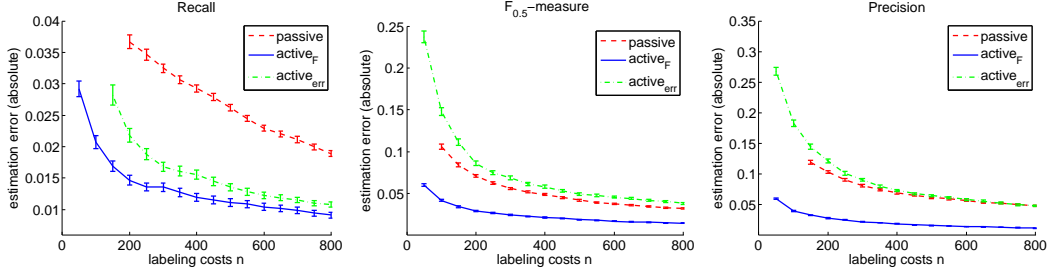

Figure 1: Spam filtering: Estimation error over labeling costs. Error bars indicate the standard error.

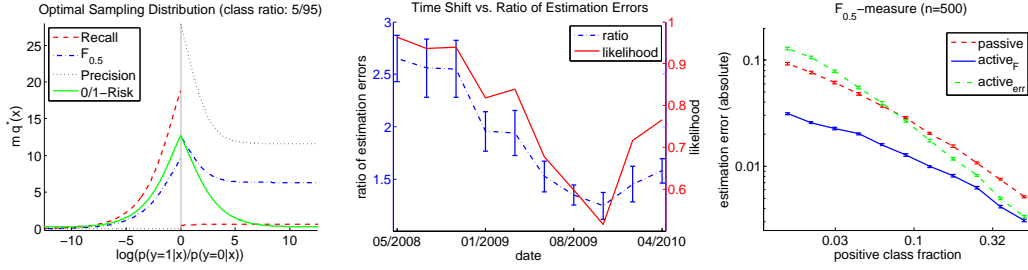

Figure 2: Spam filtering: Optimal sampling distribution $q^*$ for $F_\alpha$ over log-odds (left). Ratio of passive and active estimation error, error bars indicate standard deviation (center). Estimation error over class ratio, logarithmic scale, error bars indicate standard errors (right).

$active_F$ are at least as accurate as those of $active_{err}$, and more accurate for high $\alpha$ values. Results obtained in the digit recognition domain are consistent with these findings (see online appendix).

Figure 2 (left) shows the sampling distribution $q^*(\mathbf{x})$ for recall, precision and $F_{0.5}$-measure in the spam filtering domain as a function of the classifier's confidence, characterized by the log-odds ratio $\log \frac{p(y=1|\mathbf{x};\theta)}{p(y=0|\mathbf{x};\theta)}$. The figure also shows the optimal sampling distribution for zero-one risk as used in $active_{err}$ (denoted "0/1-Risk"). We observe that the precision estimator dismisses all examples with $f_\theta(\mathbf{x}) = 0$; this is intuitive because precision is a function of true-positive and false-positive examples only. By contrast, the recall estimator selects examples on both sides of the decision boundary, as it has to estimate both the true positive and the false negative rate. The optimal sampling distribution for zero-one risk is symmetric, it prefers instances close to the decision boundary.

**Effect of discrepancy between training and test distribution.** We keep the training set of emails fixed and move the time interval from which test instances are drawn increasingly further away into the future, thereby creating a growing gap between training and test distribution. Specifically, we divide 127,447 emails received between February 2008 and April 2010 into ten different test sets spanning approximately 2.5 months each. Figure 2 (center, red curve) shows the discrepancy between training and test distribution measured in terms of the exponentiated average log-likelihood of the test labels given the model parameters $\theta$. The likelihood at first continually decreases. It grows again for the two most recent batches; this coincides with a recent wave of text-based *vintage spam*. Figure 2 (center, blue curve) also shows the ratio of passive-to-active estimation errors $\frac{|\hat{G}_n - G|}{|\hat{G}_{n,q^*} - G|}$. A value above one indicates that the active estimate is more accurate than a passive estimate. The active estimate consistently outperforms the passive estimate; its advantage diminishes when training and test distributions diverge and the assumption of $p(y|\mathbf{x}) \approx p(y|\mathbf{x};\theta)$ becomes less accurate.

**Effect of class skew.** In the spam filtering domain we artificially sub-sampled data to different ratios of spam and non-spam emails. Figure 2 (right) shows the performance of $active_F$, $passive$, and $active_{err}$ for $F_{0.5}$ estimation as a function of class skew. We observe that $active_F$ outperforms $passive$ consistently. Furthermore, $active_F$ outperforms $active_{err}$ for imbalanced classes, while the approaches perform comparably when classes are balanced. This finding is consistent with the intuition that accuracy and $F$-measure diverge more strongly for imbalanced classes.

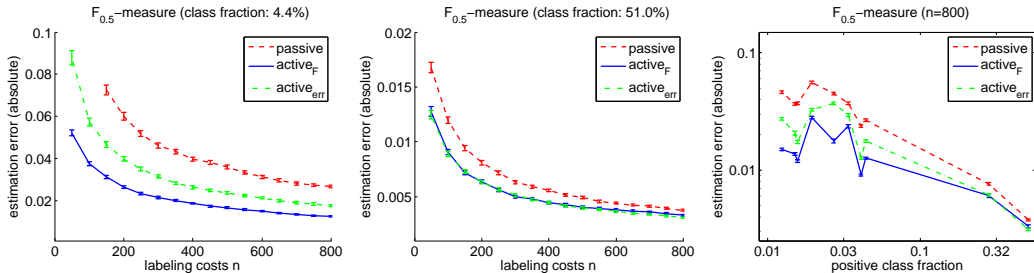

Figure 3: Text classification: Estimation error over number of labeled data for infrequent (left) and frequent (center) class. Estimation error over class ratio for all ten classes, logarithmic scale (right). Error bars indicate the standard error.

In the text classification domain we estimate the $F_{0.5}$-measure for ten one-versus-rest classifiers. Figure 3 shows the estimation error of $active_F$, $passive$, and $active_{err}$ for an infrequent class ("crude", $4.41\%$, left) and a frequent class ("earn", $51.0\%$, center). These results are representative for other frequent and infrequent classes, all results are included in the online appendix. Figure 3 (right) shows the estimation error of $active_F$, $passive$, and $active_{err}$ on all ten one-versus-rest problems as a function of the problem's class skew. We again observe that $active_F$ outperforms $passive$ consistently, and $active_F$ outperforms $active_{err}$ for strongly skewed class distributions.

## 4 Related Work

Sawade et al. [7] derive a variance-minimizing sampling distribution for risks. Their result does not cover $F$-measures. Our experimental findings show that for estimating $F$-measures their variance-minimizing sampling distribution performs worse than the sampling distributions characterized by Theorem 1, especially for skewed class distributions.

Active estimation of generalized risks can be considered to be a dual problem of active learning; in active learning, the goal of the selection process is to minimize the variance of the predictions or the variance of the model parameters, while in active evaluation the variance of the risk estimate is reduced. The variance-minimizing sampling distribution derived in Section 2.1 depends on the unknown conditional distribution $p(y|\mathbf{x})$. We use the model itself to approximate this distribution and decide on instances whose class labels are queried. This is analogous to many active learning algorithms. Specifically, Bach derives a sampling distribution for active learning under the assumption that the current model gives a good approximation to the conditional probability $p(y|\mathbf{x})$ [1]. To compensate for the bias incurred by the instrumental distribution, several active learning algorithms use importance weighting: for regression [8], exponential family models [1], or SVMs [2].

Finally, the proposed active estimation approach can be considered an instance of the general principle of importance sampling [6], which we employ in the context of generalized risk estimation.

## 5 Conclusions

$F_\alpha$-measures are defined as empirical estimates; we have shown that they are consistent estimates of a generalized risk functional which Proposition 1 identifies. Generalized risks can be estimated actively by sampling test instances from an instrumental distribution $q$. An analysis of the sources of estimation error leads to an instrumental distribution $q^*$ that minimizes estimator variance. The optimal sampling distribution depends on the unknown conditional $p(y|\mathbf{x})$; the active generalized risk estimator approximates this conditional by the model to be evaluated.

Our empirical study supports the conclusion that the advantage of active over passive evaluation is particularly strong for skewed classes. The advantage of active evaluation is also correlated to the quality of the model as measured by the model-based likelihood of the test labels. In our experiments, active evaluation consistently outperformed passive evaluation, even for the greatest divergence between training and test distribution that we could observe.

## Appendix

### Proof of Lemma 2

Let $(\mathbf{x}_1, y_1), ..., (\mathbf{x}_n, y_n)$ be drawn according to $q(\mathbf{x})p(y|\mathbf{x})$. Let $\hat{G}^0_{n,q} = \sum_{i=1}^n v_i \ell_i w_i$ and $W_n = \sum_{i=1}^n v_i w_i$ with $v_i = \frac{p(\mathbf{x}_i)}{q(\mathbf{x}_i)}$, $w_i = w(\mathbf{x}_i, y_i, f_\theta)$ and $\ell_i = \ell(f_\theta(\mathbf{x}_i), y_i)$. We note that $\mathbb{E}\left[\hat{G}^0_{n,q}\right] = nG\,\mathbb{E}\left[w_i\right]$ and $\mathbb{E}\left[W_n\right] = n\,\mathbb{E}\left[w_i\right]$. The random variables $w_1 v_1, \ldots, w_n v_n$ and $w_1 \ell_1 v_1, \ldots, w_n \ell_n v_n$ are *iid*, therefore the central limit theorem implies that $\frac{1}{n}\hat{G}^0_{n,q}$ and $\frac{1}{n}W_n$ are asymptotically normally distributed with

$$\sqrt{n}\left(\frac{1}{n}\hat{G}^0_{n,q} - G\,\mathbb{E}\left[w_i\right]\right) \stackrel{n\to\infty}{\longrightarrow} \mathcal{N}(0, \mathrm{Var}[w_i \ell_i v_i]) \tag{17}$$

$$\sqrt{n}\left(\frac{1}{n}W_n - \mathbb{E}\left[w_i\right]\right) \stackrel{n\to\infty}{\longrightarrow} \mathcal{N}(0, \mathrm{Var}[w_i v_i]) \tag{18}$$

where $\stackrel{n\to\infty}{\longrightarrow}$ denotes convergence in distribution. Application of the delta method to the function $f(x, y) = \frac{x}{y}$ yields

$$\sqrt{n}\left(\frac{\frac{1}{n}\hat{G}^0_{n,q}}{\frac{1}{n}W_n} - G\right) \stackrel{n\to\infty}{\longrightarrow} \mathcal{N}(0, \nabla f\left(G\,\mathbb{E}\left[w_i\right], \mathbb{E}\left[w_i\right]\right)^\mathsf{T} \Sigma \nabla f\left(G\,\mathbb{E}\left[w_i\right], \mathbb{E}\left[w_i\right]\right))$$

where $\nabla f$ denotes the gradient of $f$ and $\Sigma$ is the asymptotic covariance matrix of the input arguments

$$\Sigma = \begin{pmatrix} \mathrm{Var}[w_i \ell_i v_i] & \mathrm{Cov}[w_i \ell_i v_i, w_i v_i] \\ \mathrm{Cov}[w_i \ell_i v_i, w_i v_i] & \mathrm{Var}[w_i v_i] \end{pmatrix}.$$

Furthermore,

$$\nabla f\left(G\,\mathbb{E}\left[w_i\right], \mathbb{E}\left[w_i\right]\right)^\mathsf{T} \Sigma \nabla f\left(G\,\mathbb{E}\left[w_i\right], \mathbb{E}\left[w_i\right]\right)$$
$$= \mathrm{Var}\left[w_i \ell_i v_i\right] - 2G\,\mathrm{Cov}\left[w_i v_i, w_i \ell_i v_i\right] + G^2\,\mathrm{Var}\left[w_i v_i\right]$$
$$= \mathbb{E}\left[w_i^2 \ell_i^2 v_i^2\right] - 2G\,\mathbb{E}\left[w_i^2 \ell_i v_i^2\right] + G^2\,\mathbb{E}\left[w_i^2 v_i^2\right]$$
$$= \iint \left(\frac{p(\mathbf{x})}{q(\mathbf{x})}\right)^2 w(\mathbf{x}, y, f_\theta)^2 \left(\ell(f_\theta(\mathbf{x}), y) - G\right)^2 p(y|\mathbf{x})q(\mathbf{x})dy d\mathbf{x}.$$

From this, the claim follows by canceling $q(\mathbf{x})$. $\square$

### Proof of Theorem 1

To minimize the variance with respect to the function $q$ under the the normalization constraint $\int q(\mathbf{x})d\mathbf{x} = 1$ we define the Lagrangian with Lagrange multiplier $\beta$

$$\mathcal{L}\left[q, \beta\right] = \int \frac{c(\mathbf{x})}{q(\mathbf{x})}d\mathbf{x} + \beta\left(\int q(\mathbf{x})d\mathbf{x} - 1\right) = \int \underbrace{\frac{c(\mathbf{x})}{q(\mathbf{x})} + \beta q(\mathbf{x})}_{=K(q(\mathbf{x}), \mathbf{x})} d\mathbf{x} - \beta, \tag{19}$$

where $c(\mathbf{x}) = p(\mathbf{x})^2 \int w(\mathbf{x}, y, f_\theta)^2 \left(\ell(f_\theta(\mathbf{x}), y) - G\right)^2 p(y|\mathbf{x})dy$. The optimal function for the constrained problem satisfies the Euler-Lagrange equation $\frac{\partial K}{\partial q(\mathbf{x})} = -\frac{c(\mathbf{x})}{q(\mathbf{x})^2} + \beta = 0$. A solution for this Equation under the side condition is given by

$$q^*(\mathbf{x}) = \frac{\sqrt{c(\mathbf{x})}}{\int \sqrt{c(\mathbf{x})}d\mathbf{x}}. \tag{20}$$

Note that we dismiss the negative solution, since $q(\mathbf{x})$ is a probability density function. Resubstitution of $c$ in Equation 20 implies the theorem. $\square$

## Acknowledgments

We gratefully acknowledge that this work was supported by a Google Research Award. We wish to thank Michael Brückner for his help with the experiments on spam data.

## References

[1] F. Bach. Active learning for misspecified generalized linear models. In *Advances in Neural Information Processing Systems*, 2007.

[2] A. Beygelzimer, S. Dasgupta, and J. Langford. Importance weighted active learning. In *Proceedings of the International Conference on Machine Learning*, 2009.

[3] H Cramér. *Mathematical Methods of Statistics*, chapter 20. Princeton University Press, 1946.

[4] A. Frank and A. Asuncion. UCI machine learning repository, 2010.

[5] S. Geman, E. Bienenstock, and R. Doursat. Neural networks and the bias/variance dilemma. *Neural Computation*, 4:1–58, 1992.

[6] J. Hammersley and D. Handscomb. *Monte carlo methods*. Taylor & Francis, 1964.

[7] C. Sawade, N. Landwehr, S. Bickel, and T. Scheffer. Active risk estimation. In *Proceedings of the 27th International Conference on Machine Learning*, 2010.

[8] M. Sugiyama. Active learning in approximately linear regression based on conditional expectation of generalization error. *Journal of Machine Learning Research*, 7:141–166, 2006.

[9] S. Tong and D. Koller. Support vector machine active learning with applications to text classification. *Journal of Machine Learning Research*, pages 45–66, 2002.

[10] C. van Rijsbergen. *Information Retrieval*. Butterworths, 2nd edition, 1979.

[11] M. Yamada, M. Sugiyama, and T. Matsui. Semi-supervised speaker identification under covariate shift. *Signal Processing*, 90(8):2353–2361, 2010.

